# Inference for the Generalization Error

**Claude Nadeau**
CIRANO
2020, University,
Montreal, Qc, Canada, H3A 2A5
jcnadeau@altavista.net

**Yoshua Bengio**
CIRANO and *Dept. IRO*
*Université de Montréal*
Montreal, Qc, Canada, H3C 3J7
bengioy@iro.umontreal.ca

## Abstract

In order to to compare learning algorithms, experimental results reported in the machine learning litterature often use statistical tests of significance. Unfortunately, most of these tests do not take into account the variability due to the choice of training set. We perform a theoretical investigation of the variance of the cross-validation estimate of the generalization error that takes into account the variability due to the choice of training sets. This allows us to propose two new ways to estimate this variance. We show, via simulations, that these new statistics perform well relative to the statistics considered by Dietterich (Dietterich, 1998).

## 1 Introduction

When applying a learning algorithm (or comparing several algorithms), one is typically interested in estimating its generalization error. Its point estimation is rather trivial through cross-validation. Providing a variance estimate of that estimation, so that hypothesis testing and/or confidence intervals are possible, is more difficult, especially, as pointed out in (Hinton et al., 1995), if one wants to take into account the variability due to the choice of the training sets (Breiman, 1996). A notable effort in that direction is Dietterich's work (Dietterich, 1998). Careful investigation of the variance to be estimated allows us to provide new variance estimates, which turn out to perform well.

Let us first lay out the framework in which we shall work. We assume that data are available in the form $Z_1^n = \{Z_1, \ldots, Z_n\}$. For example, in the case of supervised learning, $Z_i = (X_i, Y_i) \in \mathcal{Z} \subseteq \mathbb{R}^{p+q}$, where $p$ and $q$ denote the dimensions of the $X_i$'s (inputs) and the $Y_i$'s (outputs). We also assume that the $Z_i$'s are independent with $Z_i \sim P(Z)$. Let $\mathcal{L}(D; Z)$, where $D$ represents a subset of size $n_1 \leq n$ taken from $Z_1^n$, be a function $\mathcal{Z}^{n_1} \times \mathcal{Z} \to \mathbb{R}$. For instance, this function could be the loss incurred by the decision that a learning algorithm trained on $D$ makes on a new example $Z$. We are interested in estimating $_n\mu \equiv E[\mathcal{L}(Z_1^n; Z_{n+1})]$ where $Z_{n+1} \sim P(Z)$ is independent of $Z_1^n$. Subscript $n$ stands for the size of the training set ($Z_1^n$ here). The above expectation is taken over $Z_1^n$ and $Z_{n+1}$, meaning that we are interested in the performance of an algorithm rather than the performance of the specific decision function it yields on the data at hand. According to Dietterich's taxonomy (Dietterich, 1998), we deal with problems of type 5 through 8, (evaluating learning algorithms) rather then type 1 through 4 (evaluating decision functions). We call $_n\mu$ the generalization error even though it can also represent an error difference:

- **Generalization error**
We may take

$$\mathcal{L}(D; Z) = \mathcal{L}(D; (X, Y)) = Q(F(D)(X), Y), \tag{1}$$

where $F(D)$ $(F(D) : \mathbb{R}^p \to \mathbb{R}^q)$ is the decision function obtained when training an algorithm on $D$, and $Q$ is a loss function measuring the inaccuracy of a decision. For instance, we could have $Q(\hat{y}, y) = I[\hat{y} \neq y]$, where $I[\ ]$ is the indicator function, for classification problems and $Q(\hat{y}, y) = \| \hat{y} - y \|^2$, where is $\| \cdot \|$ is the Euclidean norm, for "regression" problems. In that case $_n\mu$ is what most people call the generalization error.

• **Comparison of generalization errors**

Sometimes, we are not interested in the performance of algorithms *per se*, but instead in how two algorithms compare with each other. In that case we may want to consider

$$\mathcal{L}(D; Z) = \mathcal{L}(D;(X,Y)) = Q(F_A(D)(X),Y) - Q(F_B(D)(X),Y), \qquad (2)$$

where $F_A(D)$ and $F_B(D)$ are decision functions obtained when training two algorithms (A and B) on $D$, and $Q$ is a loss function. In this case $_n\mu$ would be a difference of generalization errors as outlined in the previous example.

The generalization error is often estimated via some form of cross-validation. Since there are various versions of the latter, we lay out the specific form we use in this paper.

• Let $S_j$ be a random set of $n_1$ distinct integers from $\{1,\ldots,n\}(n_1 < n)$. Here $n_1$ represents the size of the training set and we shall let $n_2 = n - n_1$ be the size of the test set.

• Let $S_1,\ldots S_J$ be independent such random sets, and let $S_j^c = \{1,\ldots,n\} \setminus S_j$ denote the complement of $S_j$.

• Let $Z_{S_j} = \{Z_i | i \in S_j\}$ be the training set obtained by subsampling $Z_1^n$ according to the random index set $S_j$. The corresponding test set is $Z_{S_j^c} = \{Z_i | i \in S_j^c\}$.

• Let $L(j,i) = \mathcal{L}(Z_{S_j}; Z_i)$. According to (1), this could be the error an algorithm trained on the training set $Z_{S_j}$ makes on example $Z_i$. According to (2), this could be the difference of such errors for two different algorithms.

• Let $\hat{\mu}_j = \frac{1}{K}\sum_{k=1}^K L(j, i_k^j)$ where $i_1^j,\ldots,i_K^j$ are randomly and independently drawn from $S_j^c$. Here we draw $K$ examples from the test set $Z_{S_j^c}$ with replacement and compute the average error committed. The notation does not convey the fact that $\hat{\mu}_j$ depends on $K$, $n_1$ and $n_2$.

• Let $\hat{\mu}_j^\infty = \lim_{K\to\infty} \hat{\mu}_j = \frac{1}{n_2}\sum_{i\in S_j^c} L(j,i)$ denote what $\hat{\mu}_j$ becomes as $K$ increases without bounds. Indeed, when sampling infinitely often from $Z_{S_j^c}$, each $Z_i$ $(i \in S_j^c)$ is chosen with relative frequency $\frac{1}{n_2}$, yielding the usual "average test error". The use of $K$ is just a mathematical device to make the test examples sampled independently from $S_j^c$.

Then the cross-validation estimate of the generalization error considered in this paper is

$$_{n_1}^{n_2}\hat{\mu}_J^K = \frac{1}{J}\sum_{j=1}^J \hat{\mu}_j.$$

We note that this an unbiased estimator of $_{n_1}\mu = E[\mathcal{L}(Z_1^{n_1}, Z_{n+1})]$ (not the same as $_n\mu$).

This paper is about the estimation of the variance of $_{n_1}^{n_2}\hat{\mu}_J^\infty$. We first study theoretically this variance in section 2, leading to two new variance estimators developed in section 3. Section 4 shows part of a simulation study we performed to see how the proposed statistics behave compared to statistics already in use.

## 2   Analysis of $Var[\ _{n_1}^{n_2}\hat{\mu}_J^K]$

Here we study $Var[\ _{n_1}^{n_2}\hat{\mu}_J^\infty]$. This is important to understand why some inference procedures about $_{n_1}\mu$ presently in use are inadequate, as we shall underline in section 4. This investigation also enables us to develop estimators of $Var[\ _{n_1}^{n_2}\hat{\mu}_J^\infty]$ in section 3. Before we proceed, we state the following useful lemma, proved in (Nadeau and Bengio, 1999).

**Lemma 1** *Let $U_1, \ldots, U_k$ be random variables with common mean $\beta$, common variance $\delta$ and $Cov[U_i, U_j] = \gamma$, $\forall i \neq j$. Let $\pi = \frac{\gamma}{\delta}$ be the correlation between $U_i$ and $U_j$ $(i \neq j)$. Let $\bar{U} = k^{-1} \sum_{i=1}^{k} U_i$ and $S_U^2 = \frac{1}{k-1} \sum_{i=1}^{k} (U_i - \bar{U})^2$ be the sample mean and sample variance respectively. Then $E[S_U^2] = \delta - \gamma$ and $Var[\bar{U}] = \gamma + \frac{(\delta - \gamma)}{k} = \delta \left( \pi + \frac{1 - \pi}{k} \right)$.*

To study $Var[\,_{n_1}^{n_2}\hat{\mu}_J^K]$ we need to define the following covariances.

- Let $\sigma_0 = \sigma_0(n_1) = Var[L(j, i)]$ when $i$ is randomly drawn from $S_j^c$.
- Let $\sigma_1 = \sigma_1(n_1, n_2) = Cov[L(j, i), L(j, i')]$ for $i$ and $i'$ randomly and independently drawn from $S_j^c$.
- Let $\sigma_2 = \sigma_2(n_1, n_2) = Cov[L(j, i), L(j', i')]$, with $j \neq j'$, $i$ and $i'$ randomly and independently drawn from $S_j^c$ and $S_{j'}^c$ respectively.
- Let $\sigma_3 = \sigma_3(n_1) = Cov[L(j, i), L(j, i')]$ for $i, i' \in S_j^c$ and $i \neq i'$. This is not the same as $\sigma_1$. In fact, it may be shown that

$$\sigma_1 = Cov[L(j, i), L(j, i')] = \frac{\sigma_0}{n_2} + \frac{(n_2 - 1)\sigma_3}{n_2} = \sigma_3 + \frac{\sigma_0 - \sigma_3}{n_2}. \quad (3)$$

Let us look at the mean and variance of $\hat{\mu}_j$ and $\,_{n_1}^{n_2}\hat{\mu}_J^K$. Concerning expectations, we obviously have $E[\hat{\mu}_j] = \,_{n_1}\mu$ and thus $E[\,_{n_1}^{n_2}\hat{\mu}_J^\infty] = \,_{n_1}\mu$. From Lemma 1, we have $Var[\hat{\mu}_j] = \sigma_1 + \frac{\sigma_0 - \sigma_1}{K}$ which implies

$$Var[\hat{\mu}_j^\infty] = Var[\lim_{K \to \infty} \hat{\mu}_j] = \lim_{K \to \infty} Var[\hat{\mu}_j] = \sigma_1.$$

It can also be shown that $Cov[\hat{\mu}_j, \hat{\mu}_{j'}] = \sigma_2$, $j \neq j'$, and therefore (using Lemma 1)

$$Var[\,_{n_1}^{n_2}\hat{\mu}_J^K] = \sigma_2 + \frac{Var[\hat{\mu}_j] - \sigma_2}{J} = \sigma_2 + \frac{\sigma_1 + \frac{\sigma_0 - \sigma_1}{K} - \sigma_2}{J}. \quad (4)$$

We shall often encounter $\sigma_0, \sigma_1, \sigma_2, \sigma_3$ in the future, so some knowledge about those quantities is valuable. Here's what we can say about them.

**Proposition 1** *For given $n_1$ and $n_2$, we have $0 \leq \sigma_2 \leq \sigma_1 \leq \sigma_0$ and $0 \leq \sigma_3 \leq \sigma_1$.*
**Proof** *See (Nadeau and Bengio, 1999).*

A natural question about the estimator $\,_{n_1}^{n_2}\hat{\mu}_J^K$ is how $n_1$, $n_2$, $K$ and $J$ affect its variance.

**Proposition 2** *The variance of $\,_{n_1}^{n_2}\hat{\mu}_J^K$ is non-increasing in $J$, $K$ and $n_2$.*
**Proof** *See (Nadeau and Bengio, 1999).*

Clearly, increasing $K$ leads to smaller variance because the noise introduced by sampling with replacement from the test set disappears when this is done over and over again. Also, averaging over many train/test (increasing $J$) improves the estimation of $\,_{n_1}\mu$. Finally, all things equal elsewhere ($n_1$ fixed among other things), the larger the size of the test sets, the better the estimation of $\,_{n_1}\mu$.

The behavior of $Var[\,_{n_1}^{n_2}\hat{\mu}_J^K]$ with respect to $n_1$ is unclear, but we conjecture that **in most situations it should decrease in $n_1$**. Our argument goes like this. The variability in $\,_{n_1}^{n_2}\hat{\mu}_J^K$ comes from two sources: sampling decision rules (training process) and sampling testing examples. Holding $n_2$, $J$ and $K$ fixed freezes the second source of variation as it solely depends on those three quantities, not $n_1$. The problem to solve becomes: how does $n_1$ affect the first source of variation? It is not unreasonable to say that the decision function yielded by a learning algorithm is less variable when the training set is large. We conclude that the first source of variation, and thus the total variation (that is $Var[\,_{n_1}^{n_2}\hat{\mu}_J^K]$) is decreasing in $n_1$. We advocate the use of the estimator

$$\,_{n_1}^{n_2}\hat{\mu}_J^\infty = \frac{1}{J} \sum_{j=1}^{J} \hat{\mu}_j^\infty \quad (5)$$

as it is easier to compute and has smaller variance than $\substack{n_2\\n_1}\hat{\mu}_J^K$ ($J, n_1, n_2$ held constant).

$$Var[\substack{n_2\\n_1}\hat{\mu}_J^\infty] = \lim_{K\to\infty} Var[\substack{n_2\\n_1}\hat{\mu}_J^K] = \sigma_2 + \frac{\sigma_1 - \sigma_2}{J} = \sigma_1\left(\rho + \frac{1-\rho}{J}\right). \quad (6)$$

where $\rho = \frac{\sigma_2}{\sigma_1} = Corr[\hat{\mu}_j^\infty, \hat{\mu}_{j'}^\infty]$.

# 3  Estimation of $Var[\substack{n_2\\n_1}\hat{\mu}_J^\infty]$

We are interested in estimating $\substack{n_2\\n_1}\sigma_J^2 \equiv Var[\substack{n_2\\n_1}\hat{\mu}_J^\infty]$ where $\substack{n_2\\n_1}\hat{\mu}_J^\infty$ is as defined in (5). We provide two different estimators of $Var[\substack{n_2\\n_1}\hat{\mu}_J^\infty]$. The first is simple but may have a positive or negative bias for the actual variance. The second is meant to be conservative, that is, if our conjecture of the previous section is correct, its expected value exceeds the actual variance.

**1st Method: Corrected Resampled $t$-Test.** Let us recall that $\substack{n_2\\n_1}\hat{\mu}_J^\infty = \frac{1}{J}\sum_{j=1}^J \hat{\mu}_j^\infty$. Let $\tilde{\sigma}^2$ be the sample variance of the $\hat{\mu}_j^\infty$'s. According to Lemma 1,

$$E[\tilde{\sigma}^2] = \sigma_1(1-\rho) = \frac{1-\rho}{\rho + \frac{1-\rho}{J}}\sigma_1\left(\rho + \frac{1-\rho}{J}\right) = \frac{\sigma_1\left(\rho + \frac{1-\rho}{J}\right)}{\frac{1}{J} + \frac{\rho}{1-\rho}} = \frac{Var[\substack{n_2\\n_1}\hat{\mu}_J^\infty]}{\frac{1}{J} + \frac{\rho}{1-\rho}}, \quad (7)$$

so that $\left(\frac{1}{J} + \frac{\rho}{1-\rho}\right)\tilde{\sigma}^2$ is an unbiased estimator of $Var[\substack{n_2\\n_1}\hat{\mu}_J^\infty]$. The only problem is that $\rho = \rho(n_1, n_2) = \frac{\sigma_2(n_1,n_2)}{\sigma_1(n_1,n_2)}$, the correlation between the $\hat{\mu}_j^\infty$'s, is unknown and difficult to estimate. We use a naive surrogate for $\rho$ as follows. Let us recall that $\hat{\mu}_j^\infty = \frac{1}{n_2}\sum_{i\in S_j^c}\mathcal{L}(Z_{S_j}; Z_i)$. For the purpose of building our estimator, let us make the approximation that $\mathcal{L}(Z_{S_j}; Z_i)$ depends only on $Z_i$ and $n_1$. Then it is not hard to show (see (Nadeau and Bengio, 1999)) that the correlation between the $\hat{\mu}_j^\infty$'s becomes $\frac{n_2}{n_1+n_2}$. Therefore our first estimator of $Var[\substack{n_2\\n_1}\hat{\mu}_J^\infty]$ is $\left(\frac{1}{J} + \frac{\rho_o}{1-\rho_o}\right)\tilde{\sigma}^2$ where $\rho_o = \rho_o(n_1, n_2) = \frac{n_2}{n_1+n_2}$, that is $\left(\frac{1}{J} + \frac{n_2}{n_1}\right)\tilde{\sigma}^2$. This will tend to overestimate or underestimate $Var[\substack{n_2\\n_1}\hat{\mu}_J^\infty]$ according to whether $\rho_o > \rho$ or $\rho_o < \rho$. Note that this first method basically does not require any more computations than that already performed to estimate generalization error by cross-validation.

**2nd Method: Conservative $Z$.** Our second method aims at overestimating $Var[\substack{n_2\\n_1}\hat{\mu}_J^\infty]$ which will lead to conservative inference, that is tests of hypothesis with actual size less than the nominal size. This is important because techniques currently in use have the opposite defect, that is they tend to be liberal (tests with actual size exceeding the nominal size), which is typically regarded as less desirable than conservative tests.

Estimating $\substack{n_2\\n_1}\sigma_J^2$ unbiasedly is not trivial as hinted above. However we may estimate unbiasedly $\substack{n_2\\n_1'}\sigma_J^2 = Var[\substack{n_2\\n_1'}\hat{\mu}_J^\infty]$ where $n_1' = \lfloor\frac{n}{2}\rfloor - n_2 < n_1$. Let $\substack{n_2\\n_1'}\hat{\sigma}_J^2$ be the unbiased estimator, developed below, of the above variance. We argued in the previous section that $Var[\substack{n_2\\n_1'}\hat{\mu}_J^\infty] \geq Var[\substack{n_2\\n_1}\hat{\mu}_J^\infty]$. Therefore $\substack{n_2\\n_1'}\hat{\sigma}_J^2$ will tend to overestimate $\substack{n_2\\n_1}\sigma_J^2$, that is $E[\substack{n_2\\n_1'}\hat{\sigma}_J^2] = \substack{n_2\\n_1'}\sigma_J^2 \geq \substack{n_2\\n_1}\sigma_J^2$.

Here's how we may estimate $\substack{n_2\\n_1'}\sigma_J^2$ without bias. For simplicity, assume that $n$ is even. We have to randomly split our data $Z_1^n$ into two distinct data sets, $D_1$ and $D_1^c$, of size $\frac{n}{2}$ each. Let $\hat{\mu}_{(1)}$ be the statistic of interest ($\substack{n_2\\n_1'}\hat{\mu}_J^\infty$) computed on $D_1$. This involves, among other things, drawing $J$ train/test subsets from $D_1$. Let $\hat{\mu}_{(1)}^c$ be the statistic computed on $D_1^c$. Then $\hat{\mu}_{(1)}$ and $\hat{\mu}_{(1)}^c$ are independent since $D_1$ and $D_1^c$ are independent data sets, so that $(\hat{\mu}_{(1)} - \frac{\hat{\mu}_{(1)}+\hat{\mu}_{(1)}^c}{2})^2 + (\hat{\mu}_{(1)}^c - \frac{\hat{\mu}_{(1)}+\hat{\mu}_{(1)}^c}{2})^2 = \frac{1}{2}(\hat{\mu}_{(1)} - \hat{\mu}_{(1)}^c)^2$ is an unbiased estimate of $\substack{n_2\\n_1'}\sigma_J^2$. This splitting process may be repeated $M$ times. This yields $D_m$ and $D_m^c$, with

$D_m \cup D_m^c = Z_1^n$, $D_m \cap D_m^c = \emptyset$ for $m = 1, \ldots, M$. Each split yields a pair $(\hat{\mu}_{(m)}, \hat{\mu}_{(m)}^c)$ that is such that $\frac{1}{2}(\hat{\mu}_{(m)} - \hat{\mu}_{(m)}^c)^2$ is unbiased for $\frac{n_2}{n_1}\sigma_J^2$. This allows us to use the following unbiased estimator of $\frac{n_2}{n_1}\sigma_J^2$:

$$\frac{n_2}{n_1}\hat{\sigma}_J^2 = \frac{1}{2M}\sum_{m=1}^{M}(\hat{\mu}_{(m)} - \hat{\mu}_{(m)}^c)^2. \tag{8}$$

Note that, according to Lemma 1, $Var[\frac{n_2}{n_1}\hat{\sigma}_J^2] = \frac{1}{4}Var[(\hat{\mu}_{(m)} - \hat{\mu}_{(m)}^c)^2](r + \frac{1-r}{M})$ with $r = Corr[(\hat{\mu}_{(i)} - \hat{\mu}_{(i)}^c)^2, (\hat{\mu}_{(j)} - \hat{\mu}_{(j)}^c)^2]$ for $i \neq j$. Simulations suggest that $r$ is usually close to 0, so that the above variance decreases roughly like $\frac{1}{M}$ for $M$ up to 20, say. The second method is therefore a bit more computation intensive, since requires to perform cross-validation $M$ times, but it is expected to be conservative.

## 4 Simulation study

We consider five different test statistics for the hypothesis $H_0 : {}_{n_1}\mu = \mu_0$. The first three are methods already in use in the machine learning community, the last two are the new methods we put forward. They all have the following form

$$\text{reject } H_0 \text{ if } \left|\frac{\hat{\mu} - \mu_0}{\hat{\sigma}}\right| > c. \tag{9}$$

Table 1 describes what they are [1]. We performed a simulation study to investigate the size (probability of rejecting the null hypothesis when it is true) and the power (probability of rejecting the null hypothesis when it is false) of the five test statistics shown in Table 1. We consider the problem of estimating generalization errors in the Letter Recognition classification problem (available from www.ics.uci.edu/pub/machine-learning-databases). The learning algorithms are

1. **Classification tree**
   We used the function **tree** in Splus version 4.5 for Windows. The default arguments were used and no pruning was performed. The function **predict** with option **type="class"** was used to retrieve the decision function of the tree: $F_A(Z_S)(X)$. Here the classification loss function $L_A(j, i) = I[F_A(Z_{S_j})(X_i) \neq Y_i]$ is equal to 1 whenever this algorithm misclassifies example $i$ when the training set is $S_j$; otherwise it is 0.

2. **First nearest neighbor**
   We apply the first nearest neighbor rule with a distorted distance metric to pull down the performance of this algorithm to the level of the classification tree (as in (Dietterich, 1998)). We have $L_B(j, i)$ equal to 1 whenever this algorithm misclassifies example $i$ when the training set is $S_j$; otherwise it is 0.

In addition to inference about the generalization errors ${}_{n_1}\mu_A$ and ${}_{n_1}\mu_B$ associated with those two algorithms, we also consider inference about ${}_{n_1}\mu_{A-B} = {}_{n_1}\mu_A - {}_{n_1}\mu_B = E[L_{A-B}(j, i)]$ where $L_{A-B}(j, i) = L_A(j, i) - L_B(j, i)$.

We sample, without replacement, 300 examples from the 20000 examples available in the Letter Recognition data base. Repeating this 500 times, we obtain 500 sets of data of the form $\{Z_1, \ldots, Z_{300}\}$. Once a data set $Z_1^{300} = \{Z_1, \ldots Z_{300}\}$ has been generated, we may

| Name | $\hat{\mu}$ | $\hat{\sigma}^2$ | $c$ | $\frac{Var[\hat{\mu}]}{E[\hat{\sigma}^2]}$ |
|---|---|---|---|---|
| $t$-test (McNemar) | ${}^{n_2}_{n_1}\hat{\mu}_1^\infty$ | $\frac{1}{n_2}SV(L(1,i))$ | $t_{n_2-1,1-\alpha/2}$ | $\frac{n_2\sigma_3+(\sigma_0-\sigma_3)}{\sigma_0-\sigma_3}>1$ |
| resampled $t$ | ${}^{n_2}_{n_1}\hat{\mu}_J^\infty$ | $\frac{1}{J}\tilde{\sigma}^2$ | $t_{J-1,1-\alpha/2}$ | $1+J\frac{\rho}{1-\rho}>1$ |
| Dietterich's $5\times 2$ cv | ${}^{n/2}_{n/2}\hat{\mu}_1^\infty$ | see (Dietterich, 1998) | $t_{5,1-\alpha/2}$ | ? |
| 1: conservative $Z$ | ${}^{n_2}_{n_1}\hat{\mu}_J^\infty$ | $\frac{n_2}{n_1'}\hat{\sigma}_J^2$ | $Z_{1-\alpha/2}$ | $\frac{\frac{n_2}{n_1}\sigma_J^2}{\frac{n_2}{n_1'}\sigma_J^2}<1$ |
| 2: corr. resampled $t$ | ${}^{n_2}_{n_1}\hat{\mu}_J^\infty$ | $\left(\frac{1}{J}+\frac{n_2}{n_1}\right)\tilde{\sigma}^2$ | $t_{J-1,1-\alpha/2}$ | $\frac{1+J\frac{\rho}{1-\rho}}{1+J\frac{n_2}{n_1}}$ |

Table 1: Description of five test statistics in relation to the rejection criteria shown in (9). $Z_p$ and $t_{k,p}$ refer to the quantile $p$ of the $N(0,1)$ and Student $t_k$ distribution respectively. $\tilde{\sigma}^2$ is as defined above (7) and $SV(L(1,i))$ is the sample variance of the $L(1,i)$'s involved in ${}^{n_2}_{n_1}\hat{\mu}_1^\infty$. The $\frac{Var[\hat{\mu}]}{E[\hat{\sigma}^2]}$ ratio (which comes from proper application of Lemma 1, except for Dietterich's $5\times 2$ cv and the Conservative Z) indicates if a test will tend to be conservative (ratio less than 1) or liberal (ratio greater than 1).

perform hypothesis testing based on the statistics shown in Table 1. A difficulty arises however. For a given $n$ ($n=300$ here), those methods don't aim at inference for the same generalization error. For instance, Dietterich's $5\times 2$ cv test aims at ${}_{n/2}\mu$, while the others aim at ${}_{n_1}\mu$ where $n_1$ would usually be different for different methods (e.g. $n_1=\frac{2n}{3}$ for the $t$ test statistic, and $n_1=\frac{9n}{10}$ for the resampled $t$ test statistic, for instance). In order to compare the different techniques, for a given $n$, we shall always aim at ${}_{n/2}\mu$, i.e. use $n_1=\frac{n}{2}$. However, for statistics involving ${}^{n_2}_{n_1}\hat{\mu}_J^\infty$ with $J>1$, normal usage would call for $n_1$ to be 5 or 10 times larger than $n_2$, not $n_1=n_2=\frac{n}{2}$. Therefore, for those statistics, we also use $n_1=\frac{n}{2}$ and $n_2=\frac{n}{10}$ so that $\frac{n_1}{n_2}=5$. To obtain ${}^{n/10}_{n/2}\hat{\mu}_J^\infty$ we simply throw out 40% of the data. For the conservative $Z$, we do the variance calculation as we would normally do ($n_2=\frac{n}{10}$ for instance) to obtain ${}^{n_2}_{n/2-n_2}\hat{\sigma}_J^2={}^{n/10}_{2n/5}\hat{\sigma}_J^2$. However, in the numerator we compute both ${}^{n/2}_{n/2}\hat{\mu}_J^\infty$ and ${}^{n_2}_{n/2}\hat{\mu}_J^\infty={}^{n/10}_{n/2}\hat{\mu}_J^\infty$ instead of ${}^{n_2}_{n-n_2}\hat{\mu}_J^\infty$, as explained above. Note that the rationale that led to the conservative $Z$ statistics is maintained, that is ${}^{n/10}_{2n/5}\hat{\sigma}_J^2$ overestimates both $Var[{}^{n/10}_{n/2}\hat{\mu}_J^\infty]$ and $Var[{}^{n/2}_{n/2}\hat{\mu}_J^\infty]$ : $E\left[{}^{n/10}_{2n/5}\hat{\sigma}_J^2\right]\geq Var[{}^{n/10}_{n/2}\hat{\mu}_J^\infty]\geq Var[{}^{n/2}_{n/2}\hat{\mu}_J^\infty]$.

Figure 1 shows the estimated power of different statistics when we are interested in $\mu_A$ and $\mu_{A-B}$. We estimate powers by computing the proportion of rejections of $H_0$. We see that tests based on the $t$-test or resampled $t$-test are liberal, they reject the null hypothesis with probability greater than the prescribed $\alpha=0.1$, when the null hypothesis is true. The other tests appear to have sizes that are either not significantly larger the 10% or barely so. Note that Dietterich's $5\times 2$cv is not very powerful (note that its curve has the lowest power on the extreme values of $mu_0$). To make a fair comparison of power between two curves, one should mentally align the size (bottom of the curve) of these two curves. Indeed, even the resampled $t$-test and the conservative $Z$ that throw out 40% of the data are more powerful. That is of course due to the fact that the $5\times 2$ cv method uses $J=1$ instead of $J=15$.

This is just a glimpse of a much larger simulation study. When studying the corrected resampled $t$-test and the conservative $Z$ in their natural habitat ($n_1=\frac{9n}{10}$ and $n_2=\frac{n}{10}$), we see that they are usually either right on the money in term of size, or slightly conservative. Their powers appear equivalent. The simulations were performed with $J$ up to 25 and $M$ up to 20. We found that taking $J$ greater than 15 did not improve much the power of the

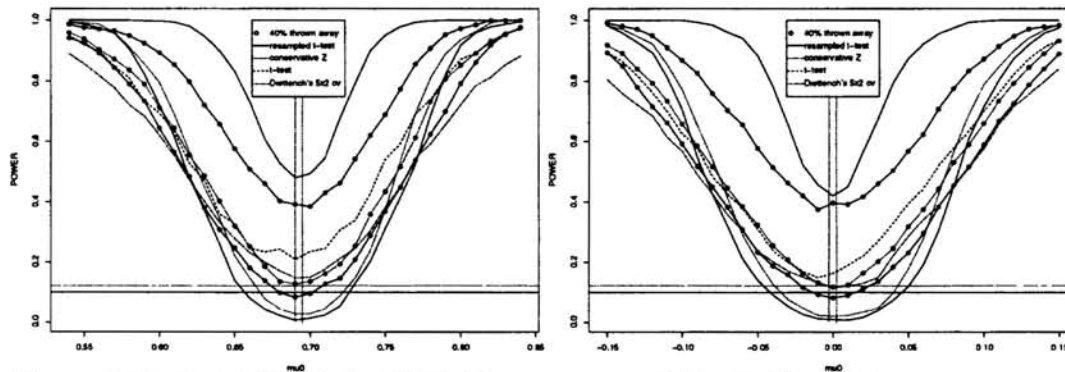

Figure 1: Powers of the tests about $H_0 : \mu_A = \mu_0$ (left panel) and $H_0 : \mu_{A-B} = \mu_0$ (right panel) at level $\alpha = 0.1$ for varying $\mu_0$. The dotted vertical lines correspond to the 95% confidence interval for the actual $\mu_A$ or $\mu_{A-B}$, therefore that is where the actual size of the tests may be read. The solid horizontal line displays the nominal size of the tests, i.e. 10%. Estimated probabilities of rejection laying above the dotted horizontal line are significatively greater than 10% (at significance level 5%). Solid curves either correspond to the resampled $t$-test or the corrected resampled $t$-test. The resampled $t$-test is the one that has ridiculously high size. Curves with circled points are the versions of the ordinary and corrected resampled $t$-test and conservative $Z$ with 40% of the data thrown away. Where it matters $J = 15$, $M = 10$ were used.

statistics. Taking $M = 20$ instead of $M = 10$ does not lead to any noticeable difference in the distribution of the conservative $Z$. Taking $M = 5$ makes the statistic slightly less conservative. See (Nadeau and Bengio, 1999) for further details.

## 5  Conclusion

This paper addresses a very important practical issue in the empirical validation of new machine learning algorithms: how to decide whether one algorithm is significantly better than another one. We argue that it is important to take into account the variability due to the choice of training set. (Dietterich, 1998) had already proposed a statistic for this purpose. We have constructed two new variance estimates of the cross-validation estimator of the generalization error. These enable one to construct tests of hypothesis and confidence intervals that are seldom liberal. Furthermore, tests based on these have powers that are unmatched by any known techniques with comparable size. One of them (corrected resampled $t$-test) can be computed without any additional cost to the usual K-fold cross-validation estimates. The other one (conservative $Z$) requires $M$ times more computation, where we found sufficiently good values of $M$ to be between 5 and 10.

## Footnotes

[1] When comparing two classifiers, (Nadeau and Bengio, 1999) show that the t-test is closely related to McNemar's test described in (Dietterich, 1998). The $5 \times 2$ cv procedure was developed in (Dietterich, 1998) with solely the comparison of classifiers in mind but may trivially be extended to other problems as shown in (Nadeau and Bengio, 1999).

## References

Breiman, L. (1996). Heuristics of instability and stabilization in model selection. *Annals of Statistics*, 24 (6):2350–2383.

Dietterich, T. (1998). Approximate statistical tests for comparing supervised classification learning algorithms. *Neural Computation*, 10 (7):1895–1924.

Hinton, G., Neal, R., Tibshirani, R., and DELVE team members (1995). Assessing learning procedures using DELVE. Technical report, University of Toronto, Department of Computer Science.

Nadeau, C. and Bengio, Y. (1999). Inference for the generalisation error. Technical Report in preparation, CIRANO.